# Robust Principal Component Analysis:
# Exact Recovery of Corrupted Low-Rank Matrices by Convex Optimization

**John Wright**, **Yigang Peng, Yi Ma**
Visual Computing Group
Microsoft Research Asia
{jowrig,v-yipe,mayi}@microsoft.com

**Arvind Ganesh, Shankar Rao**
Coordinated Science Laboratory
University of Illinois at Urbana-Champaign
{abalasu2,srrao}@uiuc.edu

## Abstract

Principal component analysis is a fundamental operation in computational data analysis, with myriad applications ranging from web search to bioinformatics to computer vision and image analysis. However, its performance and applicability in real scenarios are limited by a lack of robustness to outlying or corrupted observations. This paper considers the idealized "robust principal component analysis" problem of recovering a low rank matrix $A$ from corrupted observations $D = A + E$. Here, the corrupted entries $E$ are *unknown* and the errors can be *arbitrarily large* (modeling grossly corrupted observations common in visual and bioinformatic data), but are assumed to be sparse. We prove that most matrices $A$ can be efficiently and exactly recovered from most error sign-and-support patterns by solving a simple convex program, for which we give a fast and provably convergent algorithm. Our result holds even when the rank of $A$ grows nearly proportionally (up to a logarithmic factor) to the dimensionality of the observation space and the number of errors $E$ grows in proportion to the total number of entries in the matrix. A by-product of our analysis is the first proportional growth results for the related problem of completing a low-rank matrix from a small fraction of its entries. Simulations and real-data examples corroborate the theoretical results, and suggest potential applications in computer vision.

## 1 Introduction

The problem of finding and exploiting low-dimensional structure in high-dimensional data is taking on increasing importance in image, audio and video processing, web search, and bioinformatics, where datasets now routinely lie in thousand- or even million-dimensional observation spaces. The curse of dimensionality is in full play here: meaningful inference with limited number of observations requires some assumption that the data have low intrinsic complexity, e.g., that they are low-rank [1], sparse in some basis [2], or lie on some low-dimensional manifold [3, 4]. Perhaps the simplest useful assumption is that the observations all lie near some low-dimensional subspace. In other words, if we stack all the observations as column vectors of a matrix $M \in \mathbb{R}^{m \times n}$, the matrix should be (approximately) low rank. *Principal component analysis* (PCA) [1, 5] seeks the best (in an $\ell^2$-sense) such low-rank representation of the given data matrix. It enjoys a number of optimality properties when the data are only mildly corrupted by small noise, and can be stably and efficiently computed via the singular value decomposition.

One major shortcoming of classical PCA is its brittleness with respect to *grossly* corrupted or outlying observations [5]. Gross errors are ubiquitous in modern applications in imaging and bioinformatics, where some measurements may be arbitrarily corrupted (e.g., due to occlusion or sensor failure) or simply irrelevant to the structure we are trying to identify. A number of natural approaches to robustifying PCA have been explored in the literature. These approaches include influence function techniques [6, 7], multivariate trimming [8], alternating minimization [9], and random sampling techniques [10]. Unfortunately, none of these existing approaches yields a polynomial-time algorithm with strong performance guarantees.[1]

In this paper, we consider an idealization of the robust PCA problem, in which the goal is to recover a low-rank matrix $A$ from highly corrupted measurements $D = A + E$. The errors $E$ can be arbitrary in magnitude, but are assumed to be *sparsely* supported, affecting only a fraction of the entries of $D$. This should be contrasted with the classical setting in which the matrix $A$ is perturbed by small (but densely supported) noise. In that setting, classical PCA, computed via the singular value decomposition, remains optimal if the noise is Gaussian. Here, on the other hand, even a small fraction of large errors can cause arbitrary corruption in PCA's estimate of the low rank structure, $A$.

Our approach to robust PCA is motivated by two recent, and tightly related, lines of research. The first set of results concerns the robust solution of over-determined linear systems of equations in the presence of arbitrary, but sparse errors. These results imply that for generic systems of equations, it is possible to correct a constant fraction of arbitrary errors in polynomial time [11]. This is achieved by employing the $\ell^1$-*norm* as a convex surrogate for the highly-nonconvex $\ell^0$-norm. A parallel (and still emerging) line of work concerns the problem of computing low-rank matrix solutions to underdetermined linear equations [12, 13]. One of the most striking results concerns the exact completion of low-rank matrices from only a small fraction of their entries [13, 14, 15, 16].[2] There, a similar convex relaxation is employed, replacing the highly non-convex matrix rank with the *nuclear norm* (or sum of singular values).

The robust PCA problem outlined above combines aspects of both of these lines of work: we wish to recover a low-rank matrix from large but sparse errors. We will show that combining the solutions to the above problems (nuclear norm minimization for low-rank recovery and $\ell^1$-minimization for error correction) yields a polynomial-time algorithm for robust PCA that provably succeeds under broad conditions:

> With high probability, solving a simple convex program perfectly recovers a generic matrix $A \in \mathbb{R}^{m \times m}$ of rank as large as $C \frac{m}{\log(m)}$, from errors affecting up to a constant fraction of the $m^2$ entries.

This conclusion holds with high probability as the dimensionality $m$ increases, implying that in high-dimensional observation spaces, sparse and low-rank structures can be efficiently and exactly separated. This behavior is an example of the so-called *the blessing of dimensionality* [17].

However, this result would remain a theoretical curiosity without scalable algorithms for solving the associated convex program. To this end, we discuss how a near-solution to this convex program can be obtained relatively efficiently via proximal gradient [18, 19] and iterative thresholding techniques, similar to those proposed for matrix completion in [20, 21]. For large matrices, these algorithms are significantly faster and more scalable than general-purpose convex program solvers.

Our analysis also implies an extension of existing results for the low-rank matrix completion problem, and including the first results applicable to the proportional growth setting where the rank of the matrix grows as a constant (non-vanishing) fraction of the dimensionality:

> With overwhelming probability, solving a simple convex program perfectly recovers a generic matrix $A \in \mathbb{R}^{m \times m}$ of rank as large as $Cm$, from observations consisting of only a fraction $\rho m^2$ ($\rho < 1$) of its entries.

**Organization of this paper.** This paper is organized as follows. Section 2 formulates the robust principal component analysis problem more precisely and states the main results of this paper, placing these results in the context of existing work. The proof (available in [22]) relies on standard ideas from linear algebra and concentration of measure, but is beyond the scope of this paper. Section 3 extends existing proximal gradient techniques to give a simple, scalable algorithm for solving the robust PCA problem. In Section 4, we perform simulations and experiments corroborating the theoretical results and suggesting their applicability to real-world problems in computer vision. Finally, in Section 5, we outline several promising directions for future work.

## 2 Problem Setting and Main Results

We assume that the observed data matrix $D \in \mathbb{R}^{m \times n}$ was generated by corrupting some of the entries of a low-rank matrix $A \in \mathbb{R}^{m \times n}$. The corruption can be represented as an additive error $E \in \mathbb{R}^{m \times n}$, so that $D = A + E$. Because the error affects only a portion of the entries of $D$, $E$ is a sparse matrix. The idealized (or noise-free) robust PCA problem can then be formulated as follows:

**Problem 2.1** (Robust PCA). *Given $D = A + E$, where $A$ and $E$ are unknown, but $A$ is known to be low rank and $E$ is known to be sparse, recover $A$.*

This problem formulation immediately suggests a conceptual solution: seek the lowest rank $A$ that could have generated the data, subject to the constraint that the errors are sparse: $\|E\|_0 \leq k$. The Lagrangian reformulation of this optimization problem is

$$\min_{A,E} \; \text{rank}(A) + \gamma\|E\|_0 \quad \text{subj} \quad A + E = D. \tag{1}$$

If we could solve this problem for appropriate $\gamma$, we might hope to exactly recover the pair $(A_0, E_0)$ that generated the data $D$. Unfortunately, (1) is a highly nonconvex optimization problem, and no efficient solution is known.[3] We can obtain a tractable optimization problem by relaxing (1), replacing the $\ell^0$-norm with the $\ell^1$-norm, and the rank with the nuclear norm $\|A\|_* = \sum_i \sigma_i(A)$, yielding the following convex surrogate:

$$\min_{A,E} \; \|A\|_* + \lambda\|E\|_1 \quad \text{subj} \quad A + E = D. \tag{2}$$

This relaxation can be motivated by observing that $\|A\|_* + \lambda\|E\|_1$ is the convex envelope of $\text{rank}(A) + \lambda\|E\|_0$ over the set of $(A, E)$ such that $\max(\|A\|_{2,2}, \|E\|_{1,\infty}) \leq 1$. Moreover, recent advances in our understanding of the nuclear norm heuristic for low-rank solutions to matrix equations [12, 13] and the $\ell^1$ heuristic for sparse solutions to underdetermined linear systems [11, 24], suggest that there might be circumstances under which solving the tractable problem (2) perfectly recovers the low-rank matrix $A_0$. The main result of this paper will be to show that this is indeed true under surprisingly broad conditions. A sketch of the result is as follows: *For "almost all" pairs $(A_0, E_0)$ consisting of a low-rank matrix $A_0$ and a sparse matrix $E_0$,*

$$(A_0, E_0) = \arg\min_{A,E} \; \|A\|_* + \lambda\|E\|_1 \quad \text{subj} \quad A + E = A_0 + E_0,$$

*and the minimizer is uniquely defined.* That is, under natural probabilistic models for low-rank and sparse matrices, almost all observations $D = A_0 + E_0$ generated as the sum of a low-rank matrix $A_0$ and a sparse matrix $E_0$ can be efficiently and exactly decomposed into their generating parts by solving a convex program.[4]

Of course, this is only possible with an appropriate choice of the regularizing parameter $\lambda > 0$. From the optimality conditions for the convex program (2), it is not difficult to show that for matrices $D \in \mathbb{R}^{m \times m}$, the correct scaling is $\lambda = O\left(m^{-1/2}\right)$. Throughout this paper, unless otherwise stated, we will fix $\lambda = m^{-1/2}$. For simplicity, all of our results in this paper will be stated for square matrices $D \in \mathbb{R}^{m \times m}$, although there is little difficulty in extending them to non-square matrices.

It should be clear that not all matrices $A_0$ can be successfully recovered by solving the convex program (2). Consider, e.g., the rank-1 case where $U = [e_i]$ and $V = [e_j]$. Without additional prior knowledge, the low-rank matrix $A = USV^*$ cannot be recovered from even a single gross error. We therefore restrict our attention to matrices $A_0$ whose row and column spaces are not aligned with the standard basis. This can be done probabilistically, by asserting that the marginal distributions of $U$ and $V$ are uniform on the Stiefel manifold $\mathbb{W}_r^m$:

**Definition 2.2** (Random orthogonal model [13]). *We consider a matrix $A_0$ to be distributed according to the **random orthogonal model of rank** $r$ if its left and right singular vectors are independent uniformly distributed $m \times r$ matrices with orthonormal columns.[5] In this model, the nonzero singular values of $A_0$ can be arbitrary.*

Our model for errors is similarly natural: each entry of the matrix is independently corrupted with some probability $\rho_s$, and the signs of the corruptions are independent Rademacher random variables.

**Definition 2.3** (Bernoulli error signs and support). *We consider an error matrix $E_0$ to be drawn from the **Bernoulli sign and support model with parameter** $\rho_s$ if the entries of $\mathrm{sign}(E_0)$ are independently distributed, each taking on value 0 with probability $1 - \rho_s$, and $\pm 1$ with probability $\rho_s/2$ each. In this model, the magnitude of the nonzero entries in $E_0$ can be arbitrary.*

Our main result is the following (see [22] for a proof):

**Theorem 2.4** (Robust recovery from non-vanishing error fractions). *For any $p > 0$, there exist constants $(C_0^\star > 0, \rho_s^\star > 0, m_0)$ with the following property: if $m > m_0$, $(A_0, E_0) \in \mathbb{R}^{m \times m} \times \mathbb{R}^{m \times m}$ with the singular spaces of $A_0 \in \mathbb{R}^{m \times m}$ distributed according to the random orthogonal model of rank*

$$r \;\leq\; C_0^\star \, \frac{m}{\log(m)} \tag{3}$$

*and the signs and support of $E_0 \in \mathbb{R}^{m \times m}$ distributed according to the Bernoulli sign-and-support model with error probability $\leq \rho_s^\star$, then with probability at least $1 - Cm^{-p}$*

$$(A_0, E_0) \;=\; \arg\min \|A\|_* + \frac{1}{\sqrt{m}} \|E\|_1 \quad \mathrm{subj} \quad A + E = A_0 + E_0, \tag{4}$$

*and the minimizer is uniquely defined.*

In other words, matrices $A_0$ whose singular spaces are distributed according to the random orthogonal model can, with probability approaching one, be efficiently recovered from almost all corruption sign and support patterns without prior knowledge of the pattern of corruption.

Our line of analysis also implies strong results for the matrix completion problem studied in [13, 15, 14, 16]. We again refer the interested reader to [22] for a proof of the following result:

**Theorem 2.5** (Matrix completion in proportional growth). *There exist numerical constants $m_0$, $\rho_r^\star$, $\rho_s^\star$, $C$ all $> 0$, with the following property: if $m > m_0$ and $A_0 \in \mathbb{R}^{m \times m}$ is distributed according to the random orthogonal model of rank*

$$r \;\leq\; \rho_r^\star m, \tag{5}$$

*and $\Upsilon \subset [m] \times [m]$ is an independently chosen subset of $[m] \times [m]$ in which the inclusion of each pair $(i, j)$ is an independent $\mathrm{Bernoulli}(1 - \rho_s)$ random variable with $\rho_s \leq \rho_s^\star$, then with probability at least $1 - \exp(-Cm)$,*

$$A_0 \;=\; \arg\min \|A\|_* \quad \mathrm{subj} \quad A(i, j) = A_0(i, j) \quad \forall \; (i, j) \in \Upsilon, \tag{6}$$

*and the minimizer is uniquely defined.*

**Relationship to existing work.** Contemporaneous results due to [25] show that for $A_0$ distributed according to the random orthogonal model, and $E_0$ with Bernoulli support, correct recovery occurs with high probability provided

$$\|E_0\|_0 \;\leq\; C\, m^{1.5} \log(m)^{-1} \max(r, \log m)^{-1/2}. \tag{7}$$

This is an interesting result, especially since it makes no assumption on the signs of the errors. However, even for constant rank $r$ it guarantees correction of only a vanishing fraction $o(m^{1.5}) \ll$

$m^2$ of errors. In contrast, our main result, Theorem 2.4, states that even if $r$ grows proportional to $m/\log(m)$, non-vanishing fractions of errors are corrected with high probability. Both analyses start from the optimality condition for the convex program (2). The key technical component of this improved result is a probabilistic analysis of an iterative refinement technique for producing a dual vector that certifies optimality of the pair $(A_0, E_0)$. This approach extends techniques used in [11, 26], with additional care required to handle an operator norm constraint arising from the presence of the nuclear norm in (2). For further details we refer the interested reader to [22].

Finally, while Theorem 2.5 is not the main focus of this paper, it is interesting in light of results by [15]. That work proves that in the probabilistic model considered here, a generic $m \times m$ rank-$r$ matrix can be efficiently and exactly completed from a subset of only

$$Cmr \log^8(m) \tag{8}$$

entries. For $r > \frac{m}{\mathrm{polylog}(m)}$, this bound exceeds the number $m^2$ of possible observations. A similar result for spectral methods [14] gives exact completion from $O(m \log(m))$ measurements when $r = O(1)$. In contrast, our Theorem 2.5 implies that for certain scenarios with $r$ as large as $\rho_r m$, the matrix can be completed from a subset of $(1 - \rho_s)m^2$ entries. For matrices of large rank, this is a significant extension of [15]. However, our result does not supersede (8) for smaller ranks.

# 3  Scalable Optimization for Robust PCA

There are a number of possible approaches to solving the robust PCA semidefinite program (2). For small problem sizes, interior point methods offer superior accuracy and convergence rates. However, off-the-shelf interior point solvers become impractical for data matrices larger than about $70 \times 70$, due to the $O(m^6)$ complexity of solving for the step direction. For the experiments in this paper we use an alternative first-order method based on the proximal gradient approach of [18],[6] which we briefly introduce here. For further discussion of this approach, as well as alternatives based on duality, please see [27]. This algorithm solves a slightly relaxed version of (2), in which the equality constraint is replaced with a penalty term:

$$\min \ \mu\|A\|_* + \lambda\mu\|E\|_1 + \tfrac{1}{2}\|D - A - E\|_F^2. \tag{9}$$

Here, $\mu$ is a small constant; as $\mu \searrow 0$, the solutions to (9) approach the solution set of (2).

The approach of [18] minimizes functions of this type by forming separable quadratic approximations to the data fidelity term $\|D - A - E\|_F^2$ at a special set of points $(\tilde{A}_k, \tilde{E}_k)$ that are conspicuously chosen to obtain a convergence rate of $O\left(k^{-2}\right)$. The solutions to these subproblems,

$$A_{k+1} = \arg\min_A \ \mu\|A\|_* + \left\| A - \left( \tilde{A}_k - \tfrac{1}{4}\nabla_A \|D - A - E\|_F^2 \big|_{\tilde{A}_k, \tilde{E}_k} \right) \right\|_F^2, \tag{10}$$

$$E_{k+1} = \arg\min_E \ \lambda\mu\|E\|_1 + \left\| E - \left( \tilde{E}_k - \tfrac{1}{4}\nabla_E \|D - A - E\|_F^2 \big|_{\tilde{A}_k, \tilde{E}_k} \right) \right\|_F^2, \tag{11}$$

can be efficiently computed via the soft thresholding operator (for $E$) and the singular value thresholding operator (for $A$, see [20]). We terminate the iteration when the subgradient

$$\left( \tilde{A}_k - A_{k+1} + E_{k+1} - \tilde{E}_k, \ \tilde{E}_k - E_{k+1} + A_{k+1} - \tilde{A}_k \right)$$

$$\in \ \partial\left( \mu\|A\|_* + \lambda\mu\|E\|_1 + \tfrac{1}{2}\|D - A - E\|_F^2 \right)\Big|_{A_{k+1}, E_{k+1}}$$

has sufficiently small Frobenius norm.[7] In practice, convergence speed is dramatically improved by employing a continuation strategy in which $\mu$ starts relatively large and then decreases geometrically at each iteration until reaching a lower bound, $\bar{\mu}$ (as in [21]).

The entire procedure is summarized as Algorithm 1 below. We encourage the interested reader to consult [18] for a more detailed explanation of the choice of the proximal points $(\tilde{A}_k, \tilde{E}_k)$, as well as a convergence proof ([18] Theorem 4.1). As we will see in the next section, in practice the total number of iterations is often as small as 200. Since the dominant cost of each iteration is computing the singular value decomposition, this means that it is often possible to obtain a provably robust PCA with only a constant factor more computational resources than required for conventional PCA.

**Algorithm 1: Robust PCA via Proximal Gradient with Continuation**

---

1: **Input:** Observation matrix $D \in \mathbb{R}^{m \times n}$, weight $\lambda$.

2: $A_0, A_{-1} \leftarrow 0$, $E_0, E_{-1} \leftarrow 0$, $t_0, t_{-1} \leftarrow 1$, $\mu_0 \leftarrow .99\|D\|_{2,2}$, $\bar{\mu} \leftarrow 10^{-5}\mu_0$.

3: **while** not converged

4:    $\tilde{A}_k \leftarrow A_k + \frac{t_{k-1}-1}{t_k}\left(A_k - A_{k-1}\right)$, $\tilde{E}_k \leftarrow E_k + \frac{t_{k-1}-1}{t_k}\left(E_k - E_{k-1}\right)$.

5:    $Y_k^A \leftarrow \tilde{A}_k - \frac{1}{2}\left(\tilde{A}_k + \tilde{E}_k - D\right)$.

6:    $(U, S, V) \leftarrow \text{svd}(Y_k^A)$, $A_{k+1} \leftarrow U\left[S - \frac{\mu}{2}\mathtt{I}\right]_+ V^*$.

7:    $Y_k^E \leftarrow \tilde{E}_k - \frac{1}{2}\left(\tilde{A}_k + \tilde{E}_k - D\right)$.

8:    $E_{k+1} \leftarrow \text{sign}[Y_k^E] \circ \left[|Y_k^E| - \frac{\lambda\mu}{2}11^*\right]_+$.

9:    $t_{k+1} \leftarrow \frac{1+\sqrt{1+4t_k^2}}{2}$, $\mu \leftarrow \max(.9\mu, \bar{\mu})$.

10: **end while**

11: **Output:** $A, E$.

---

## 4  Simulations and Experiments

In this section, we first perform simulations corroborating our theoretical results and clarifying their implications. We then sketch two computer vision applications involving the recovery of intrinsically low-dimensional data from gross corruption: background estimation from video and face subspace estimation under varying illumination. [8]

**Simulation: proportional growth.**  We first demonstrate the exactness of the convex programming heuristic, as well as the efficacy of Algorithm 1, on random matrix examples of increasing dimension. We generate $A_0$ as a product of two independent $m \times r$ matrices whose elements are i.i.d. $\mathcal{N}(0,1)$ random variables. We generate $E_0$ as a sparse matrix whose support is chosen uniformly at random, and whose non-zero entries are independent and uniformly distributed in the range $[-500, 500]$. We apply the proposed algorithm to the matrix $D \doteq A_0 + E_0$ to recover $\hat{A}$ and $\hat{E}$. The results are presented in Table 1. For these experiments, we choose $\lambda = m^{-1/2}$. We observe that the proposed algorithm is successful in recovering $A_0$ even when $10\%$ of its entries are corrupted.

| $m$ | $\text{rank}(A_0)$ | $\|E_0\|_0$ | $\frac{\|\hat{A}-A_0\|_F}{\|A_0\|_F}$ | $\text{rank}(\hat{A})$ | $\|\hat{E}\|_0$ | #iterations | time (s) |
|---|---|---|---|---|---|---|---|
| 100 | 5 | 500 | $3.0 \times 10^{-4}$ | 5 | 506 | 104 | 1.6 |
| 200 | 10 | 2,000 | $2.1 \times 10^{-4}$ | 10 | 2,012 | 104 | 7.9 |
| 400 | 20 | 8,000 | $1.4 \times 10^{-4}$ | 20 | 8,030 | 104 | 64.8 |
| 800 | 40 | 32,000 | $9.9 \times 10^{-5}$ | 40 | 32,062 | 104 | 531.6 |
| 100 | 5 | 1,000 | $3.1 \times 10^{-4}$ | 5 | 1,033 | 108 | 1.6 |
| 200 | 10 | 4,000 | $2.3 \times 10^{-4}$ | 10 | 4,042 | 107 | 8.0 |
| 400 | 20 | 16,000 | $1.6 \times 10^{-4}$ | 20 | 16,110 | 107 | 66.7 |
| 800 | 40 | 64,000 | $1.2 \times 10^{-4}$ | 40 | 64,241 | 106 | 542.8 |

Table 1: **Proportional growth.** Here the rank of the matrix grows in proportion ($5\%$) to the dimensionality $m$; and the number of corrupted measurements grows in proportion to the number of entries $m^2$, top $5\%$ and bottom $10\%$, respectively. The time reported is for Matlab implementation run on a 2.8 GHz MacBook Pro.

**Simulation: phase transition w.r.t. rank and error sparsity.**  We next examine how the rank of $A$ and the proportion of errors in $E$ affect the performance our algorithm. We fix $m = 200$, and vary $\rho_r \doteq \frac{\text{rank}(A_0)}{m}$ and the error probability $\rho_s$ between 0 and 1. For each $\rho_r, \rho_s$ pair, we generate 10 pairs $(A_0, E_0)$ as in the above experiment. We deem $(A_0, E_0)$ successfully recovered

if the recovered $\hat{A}$ satisfies $\frac{\|\hat{A}-A_0\|_F}{\|A_0\|_F} < 0.01$. Figure 1 (left) plots the fraction of correct recoveries. White denotes perfect recovery in all experiments, and black denotes failure for all experiments. We observe that there is a relatively sharp *phase transition* between success and failure of the algorithm roughly above the line $\rho_r + \rho_s = 0.35$. To verify this behavior, we repeat the experiment, but only vary $\rho_r$ and $\rho_s$ between $0$ and $0.4$ with finer steps. These results, seen in Figure 1 (right), show that phase transition remains fairly sharp even at higher resolution.

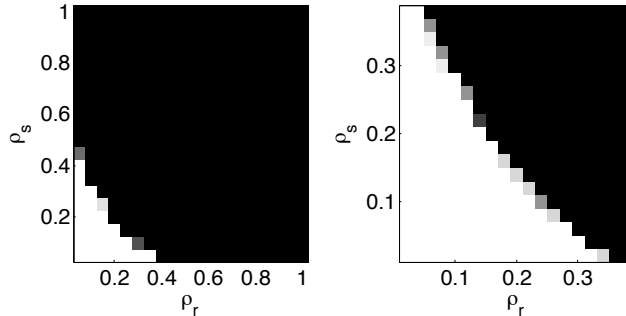

Figure 1: **Phase transition** wrt rank and error sparsity. Here, $\rho_r = \mathrm{rank}(A)/m$, $\rho_s = \|E\|_0/m^2$. Left: $(\rho_r, \rho_s) \in [0,1]^2$. Right: $(\rho_r, \rho_s) \in [0, 0.4]^2$.

**Experiment: background modeling from video.** Background modeling or subtraction from video sequences is a popular approach to detecting activity in the scene, and finds application in video surveillance from static cameras. Background estimation is complicated by the presence of foreground objects such as people, as well as variability in the background itself, for example due to varying illumination. In many cases, however, it is reasonable to assume that these background variations are low-rank, while the foreground activity is spatially localized, and therefore sparse. If the individual frames are stacked as columns of a matrix $D$, this matrix can be expressed as the sum of a low-rank background matrix and a sparse error matrix representing the activity in the scene. We illustrate this idea using two examples from [29] (see Figures 2). In Figure 2(a)-(c), the video sequence consists of 200 frames of a scene in an airport. There is no significant change in illumination in the video, but a lot of activity in the foreground. We observe that our algorithm is very effective in separating the background from the activity. In Figure 2(d)-(f), we have 550 frames from a scene in a lobby. There is little activity in the video, but the illumination changes drastically towards the end of the sequence. We see that our algorithm is once again able to recover the background, irrespective of the illumination change.

**Experiment: removing shadows and specularities from face images.** Face recognition is another domain in computer vision where low-dimensional linear models have received a great deal of attention, mostly due to the work of [30]. The key observation is that under certain idealized circumstances, images of the same face under varying illumination lie near an approximately nine-dimensional linear subspace known as the *harmonic plane*. However, since faces are neither perfectly convex nor Lambertian, face images taken under directional illumination often suffer from self-shadowing, specularities, or saturations in brightness.

Given a matrix $D$ whose columns represent well-aligned training images of a person's face under various illumination conditions, our Robust PCA algorithm offers a principled way of removing such spatially localized artifacts. Figure 3 illustrates the results of our algorithm on images from subsets 1-3 of the Extended Yale B database [31]. The proposed algorithm algorithm removes the specularities in the eyes and the shadows around the nose region. This technique is potentially useful for pre-processing training images in face recognition systems to remove such deviations from the low-dimensional linear model.

## 5 Discussion and Future Work

Our results give strong theoretical and empirical evidences for the efficacy of using convex programming to recover low-rank matrices from corrupted observations. However, there remain many fascinating open questions in this area. From a mathematical perspective, it would be interesting to

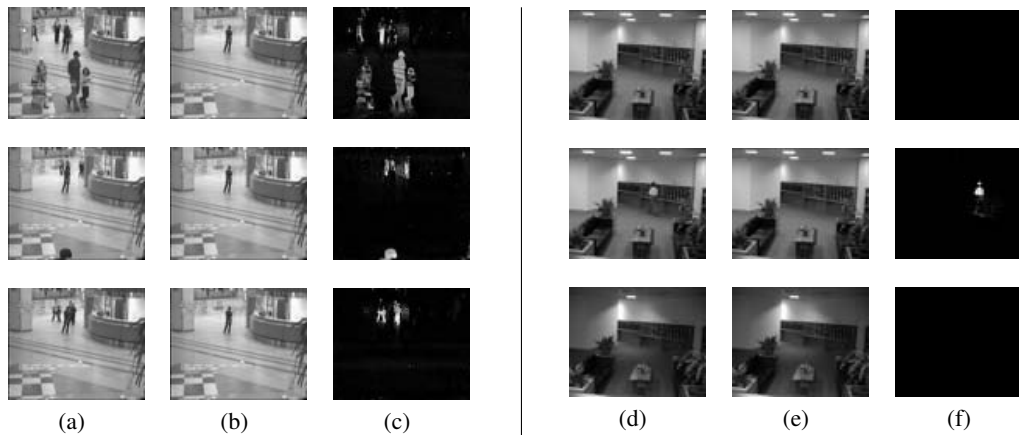

<div align="center">(a)      (b)      (c)             (d)      (e)      (f)</div>

Figure 2: **Background modeling.** (a) Video sequence of a scene in an airport. The size of each frame is $72 \times 88$ pixels, and a total of 200 frames were used. (b) Static background recovered by our algorithm. (c) Sparse error recovered by our algorithm represents activity in the frame. (d) Video sequence of a lobby scene with changing illumination. The size of each frame is $64 \times 80$ pixels, and a total of 550 frames were used. (e) Static background recovered by our algorithm. (f) Sparse error. The background is correctly recovered even when the illumination in the room changes drastically in the frame on the last row.

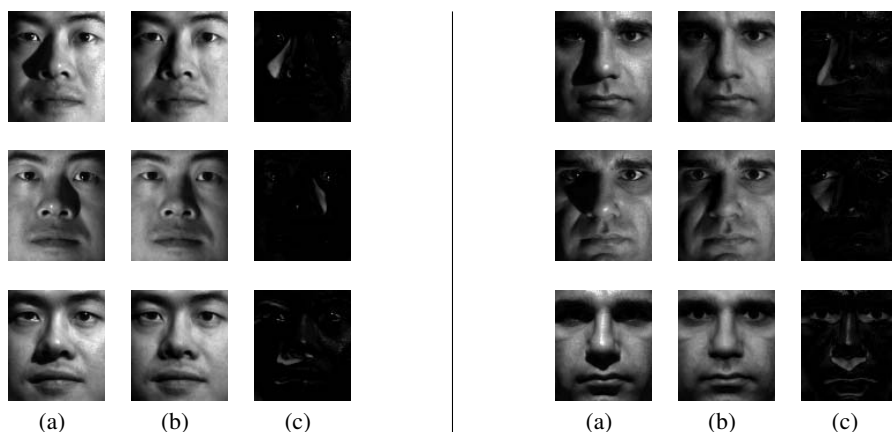

<div align="center">(a)      (b)      (c)             (a)      (b)      (c)</div>

Figure 3: **Removing shadows and specularities from face images.** (a) Cropped and aligned images of a person's face under different illuminations from the Extended Yale B database. The size of each image is $96 \times 84$ pixels, a total of 31 different illuminations were used for each person. (b) Images recovered by our algorithm. (c) The sparse errors returned by our algorithm correspond to specularities in the eyes, shadows around the nose region, or brightness saturations on the face.

know if it is possible to remove the logarithmic factor in our main result. The phase transition experiment in Section 4 suggests that convex programming actually succeeds even for $\mathrm{rank}(A_0) < \rho_r m$ and $\|E_0\|_0 < \rho_s m^2$, where $\rho_r$ and $\rho_s$ are sufficiently small positive constants. Another interesting and important question is whether the recovery is stable in the presence of small dense noise. That is, suppose we observe $D = A_0 + E_0 + Z$, where $Z$ is a noise vector of small $\ell^2$-norm (e.g., Gaussian noise). A natural approach is to now minimize $\|A\|_* + \lambda\|E\|_1$, subject to a relaxed constraint $\|D - A - E\|_F \leq \varepsilon$. For matrix completion, [16] showed that a similar relaxation gives stable recovery – the error in the solution is proportional to the noise level. Finally, while this paper has sketched several examples on visual data, we believe that this powerful new tool pertains to a wide range of high-dimensional data, for example in bioinformatics and web search.

## Footnotes

*For more information, see http://perception.csl.illinois.edu/matrix-rank/home.html. This work was partially supported by NSF IIS 08-49292, NSF ECCS 07-01676, and ONR N00014-09-1-0230.

[1] Random sampling approaches guarantee near-optimal estimates, but have complexity exponential in the rank of the matrix $A_0$. Trimming algorithms have comparatively lower computational complexity, but guarantee only locally optimal solutions.

[2] A major difference between robust PCA and low-rank matrix completion is that here we do not know which entries are corrupted, whereas in matrix completion the support of the missing entries is given.

[3]In a sense, this problem subsumes both the low rank matrix completion problem and the $\ell^0$-minimization problem, both of which are NP-hard and hard to approximate [23].

[4]Notice that this is not an "equivalence" result for (1) and (2) – rather than asserting that the solutions of these two problems are equal with high probability, we directly prove that the convex program correctly decomposes $D = A_0 + E_0$ into $(A_0, E_0)$. A natural conjecture, however, is that under the conditions of our main result, $(A_0, E_0)$ is also the solution to (1) for some choice of $\gamma$.

[5]I.e., distributed according to the Haar measure on the Stiefel manifold $\mathbb{W}_r^m$.

[6] That work is similar in spirit to the work of [19], and has also applied to matrix completion in [21].

[7] More precisely, as suggested in [21], we terminate when the norm of this subgradient is less than $2\max(1, \|(A_{k+1}, E_{k+1})\|_F) \times \tau$. In our experiments, we set $\tau = 10^{-7}$.

[8]Here, we use these intuitive examples and data illustrate how our algorithm can be used as a simple, general tool to effectively separate low-dimensional and sparse structures occurring in real visual data. Appropriately harnessing additional structure (e.g., the spatial coherence of the error [28]) may yield even more effective algorithms.

# References

[1] C. Eckart and G. Young. The approximation of one matrix by another of lower rank. *Psychometrika*, 1(3):211–218, 1936.

[2] S. Chen, D. Donoho, and M. Saunders. Atomic decomposition by basis pursuit. *SIAM Review*, 43(1):129–159, 2001.

[3] J. Tenenbaum, V. de Silva, and J. Langford. A global geometric framework for nonlinear dimensionality reduction. *Science*, 290(5500):2319–2323, 2000.

[4] M. Belkin and P. Niyogi. Laplacian eigenmaps for dimensionality reduction and data representation. *Neural Computation*, 15(6):1373–1396, 2003.

[5] I. Jolliffe. *Principal Component Analysis*. Springer-Verlag, New York, New York, 1986.

[6] P. Huber. *Robust Statistics*. Wiley, New York, New York, 1981.

[7] F. De La Torre and M. Black. A framework for robust subspace learning. *IJCV*, 54(1-3):117–142, 2003.

[8] R. Gnanadesikan and J. Kettenring. Robust estimates, residuals, and outlier detection with multiresponse data. *Biometrics*, 28(1):81–124, 1972.

[9] Q. Ke and T. Kanade. Robust l1 norm factorization in the presence of outliers and missing data by alternative convex programming. In *CVPR*, 2005.

[10] M. Fischler and R. Bolles. Random sample consensus: A paradigm for model fitting with applications to image analysis and automated cartography. *Communications of the ACM*, 24(6):381–385, 1981.

[11] E. Candès and T. Tao. Decoding by linear programming. *IEEE Trans. Info. Thy.*, 51(12):4203–4215, 2005.

[12] B. Recht, M. Fazel, and P. Parillo. Guaranteed minimum rank solution of matrix equations via nuclear norm minimization. *SIAM Review, submitted for publication*.

[13] E. Candes and B. Recht. Exact matrix completion via convex optimzation. *Foundations of Computational Mathematics, to appear*.

[14] A. Montanari R. Keshavan and S. Oh. Matrix completion from a few entries. preprint, 2009.

[15] E. Candes and T. Tao. The power of convex relaxation: Near-optimal matrix completion. *IEEE Transactions on Information Theory, submitted for publication*.

[16] E. Candes and Y. Plan. Matrix completion with noise. *Proceedings of the IEEE, to appear*.

[17] D. Donoho. High-dimensional data analysis: The curses and blessings of dimensionality. *AMS Math Challenges Lecture*, 2000.

[18] A. Beck and M. Teboulle. A fast iterative shrinkage-thresholding algorithm for linear inverse problems. *SIAM Journal on Imaging Science*, (1):183–202, 2009.

[19] Y. Nesterov. Smooth minimization of non-smooth functions. *Mathematical Programming*, 103(1):127–152, 2005.

[20] J. Cai, E. Candes, and Z. Shen. A singular value thresholding algorithm for matrix completion. preprint, `http://arxiv.org/abs/0810.3286`, 2008.

[21] K.-C. Toh and S. Yun. An accelerated proximal gradient algorithm for nuclear norm regularized least squares problems. preprint, `http://math.nus.edu.sg/~matys/apg.pdf`, 2009.

[22] J. Wright, A. Ganesh, S. Rao, and Y. Ma. Robust principal component analysis: Exact recovery of corrupted low-rank matrices via convex optimization. *Journal of the ACM, submitted for publication*.

[23] E. Amaldi and V. Kann. On the approximability of minimizing nonzero variables or unsatisfied relations in linear systems. *Theoretical Computer Science*, 209(2):237–260, 1998.

[24] D. Donoho. For most large underdetermined systems of linear equations the minimal $l_1$-norm solution is also the sparsest solution. *Communications on Pure and Applied Mathematics*, 59(6):797–829, 2006.

[25] V. Chandrasekaran, S. Sanghavi, P. Parrilo, and A. Willsky. Sparse and low-rank matrix decompositions. In *IFAC Symposium on System Identification*, 2009.

[26] J. Wright and Y. Ma. Dense error correction via $\ell^1$-minimization. *IEEE Transactions on Information Theory,* to appear.

[27] Z. Lin, A. Ganesh, J. Wright, M. Chen, L. Wu, and Y. Ma. Fast convex optimization algorithms for exact recovery of a corrupted low-rank matrix. *SIAM Journal on Optimization, submitted for publication*.

[28] V. Cevher, , M. F. Duarte, C. Hegde, and R. G. Baraniuk. Sparse signal recovery using markov random fields. In *NIPS*, 2008.

[29] L. Li, W. Huang, I. Gu, and Q. Tian. Statistical modeling of complex backgrounds for foreground object detection. *IEEE Transactions on Image Processing*, 13(11), 2004.

[30] R. Basri and D. Jacobs. Lambertian reflection and linear subspaces. *IEEE Trans. PAMI*, 25(3):218–233, 2003.

[31] A. Georghiades, P. Belhumeur, and D. Kriegman. From few to many: Illumination cone models for face recognition under variable lighting and pose. *IEEE Trans. PAMI*, 23(6):643–660, 2001.

